# Pictorial Structures for Molecular Modeling: Interpreting Density Maps

**Frank DiMaio, Jude Shavlik**
Department of Computer Sciences
University of Wisconsin-Madison
*{dimaio,shavlik}@cs.wisc.edu*

**George Phillips**
Department of Biochemistry
University of Wisconsin-Madison
*phillips@biochem.wisc.edu*

## Abstract

X-ray crystallography is currently the most common way protein structures are elucidated. One of the most time-consuming steps in the crystallographic process is interpretation of the electron density map, a task that involves finding patterns in a three-dimensional picture of a protein. This paper describes DEFT (DEFormable Template), an algorithm using *pictorial structures* to build a flexible protein model from the protein's amino-acid sequence. *Matching* this pictorial structure into the density map is a way of automating density-map interpretation. Also described are several extensions to the pictorial structure matching algorithm necessary for this automated interpretation. DEFT is tested on a set of density maps ranging from 2 to 4Å resolution, producing root-mean-squared errors ranging from 1.38 to 1.84Å.

## 1 Introduction

An important question in molecular biology is *what is the structure of a particular protein?* Knowledge of a protein's unique conformation provides insight into the mechanisms by which a protein acts. However, no algorithm exists that accurately maps sequence to structure, and one is forced to use "wet" laboratory methods to elucidate the structure of proteins. The most common such method is x-ray crystallography, a rather tedious process in which x-rays are shot through a crystal of purified protein, producing a pattern of spots (or *reflections*) which is processed, yielding an *electron density map*. The density map is analogous to a three-dimensional image of the protein. The final step of x-ray crystallography – referred to as *interpreting* the map – involves fitting a complete molecular model (that is, the position of each atom) of the protein into the map. Interpretation is typically performed by a crystallographer using a time-consuming manual process. With large research efforts being put into high-throughput structural genomics, accelerating this process is important. We investigate speeding the process of x-ray crystallography by automating this time-consuming step.

When interpreting a density map, the amino-acid sequence of the protein is known in advance, giving the complete topology of the protein. However, the intractably large conformational space of a protein – with hundreds of amino acids and thousands of atoms – makes automated map interpretation challenging. A few groups have attempted automatic interpretation, with varying success [1,2,3,4].

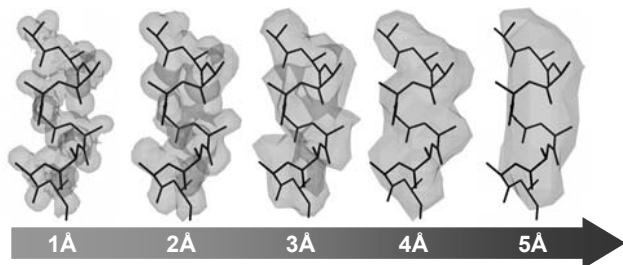

**Figure 1:** This graphic illustrates density map quality at various resolutions. All resolutions depict the same alpha helix structure

Confounding the problem are several sources of error that make automated interpretation extremely difficult. The primary source of difficulty is due to the crystal only diffracting to a certain extent, eliminating higher frequency components of the density map. This produces an overall blurring effect evident in the density map. This blurring is quantified as the *resolution* of the density map and is illustrated in Figure 1. Noise inherent in data collection further complicates interpretation. Given minimal noise and sufficiently good resolution – about 2.3Å or less – automated density map interpretation is essentially solved [1]. However, in poorer quality maps, interpretation is difficult and inaccurate, and other automated approaches have failed.

The remainder of the paper describes DEFT (*DEFormable Template*), our computational framework for building a flexible three-dimensional model of a molecule, which is then used to locate patterns in the electron density map.

## 2  Pictorial structures

Pictorial structures model classes of objects as a single flexible template. The template represents the object class as a collection of parts linked in a graph structure. Each edge defines a *relationship* between the two parts it connects. For example, a pictorial structure for a face may include the parts "*left eye*" and "*right eye*." Edges connecting these parts could enforce the constraint that the *left eye* is adjacent to the *right eye*. A dynamic programming (DP) matching algorithm of Felzenszwalb and Huttenlocher (hereafter referred to as the *F-H matching algorithm*) [5] allows pictorial structures to be quickly matched into a two-dimensional image. The matching algorithm finds the globally optimal position and orientation of each part in the pictorial structure, assuming *conditional independence* on the position of each part given its neighbors.

Formally, we represent the pictorial structure as a graph $\mathbf{G} = \mathbf{(V,E)}$, $\mathbf{V} = \{v_1, v_2, \ldots, v_n\}$ the set of parts, and edge $e_{ij} \in \mathbf{E}$ connecting neighboring parts $v_i$ and $v_j$ if an explicit dependency exists between the configurations of the corresponding parts. Each part $v_i$ is assigned a configuration $l_i$ describing the part's *position* and *orientation* in the image. We assume Markov independence: the probability distribution over a part's configurations is conditionally independent of every other part's configuration, given the configuration of all the part's neighbors in the graph. We assign each *edge* a deformation cost $\mathbf{d}_{ij}(l_i, l_j)$, and each *part* a "mismatch" cost $\mathbf{m}_i(l_i, I)$. These functions are the negative log likelihoods of a part (or pair of parts) taking a specified configuration, given the pictorial structure model.

The matching algorithm places the model into the image using maximum-likelihood. That is, it finds the configuration $L$ of parts in model $\Theta$ in image $I$ maximizing

$$P(L|I,\Theta) \propto P(I|L,\Theta)P(L|\Theta) = \frac{1}{Z}\left(\exp\left(\sum_{v_i \in \mathbf{V}} \mathrm{m}_i(l_i, \mathbf{I})\right) \cdot \exp\left(\sum_{(v_i, v_j) \in \mathbf{E}} \mathrm{m}_i(l_i, \mathbf{I})\right)\right) \tag{1}$$

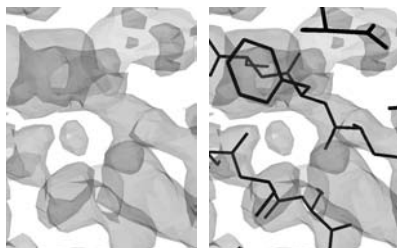

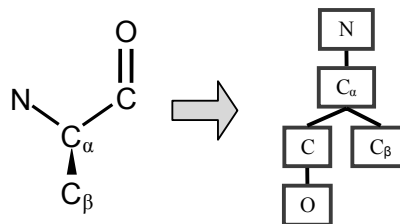

**Figure 2.** An "interpreted" density map. The right figure shows the arrangement of atoms that generated the observed density.

**Figure 3.** An example of the construction of a pictorial structure model given an amino acid.

By monotonicity of exponentiation, this minimizes $\sum_{v_i \in \mathbf{V}} m_i(l_i, \mathbf{I}) + \sum_{(v_i, v_j) \in \mathbf{E}} d_{ij}(l_i, l_j)$. The F-H matching algorithm places several additional limitations on the pictorial structure. The object's graph must be *tree structured* (cyclic constraints are not allowed), and the deformation cost function must take the form $\left\| \mathbf{T_{ij}}(l_i) - \mathbf{T_{ji}}(l_j) \right\|$, where $\mathbf{T_{ij}}$ and $\mathbf{T_{ji}}$ are arbitrary functions and $\|\cdot\|$ is some norm (e.g. Euclidian distance).

## 3   Building a flexible atomic model

Given a three-dimensional map containing a large molecule and the topology (i.e., for proteins, the amino-acid sequence) of that molecule, our task is to determine the Cartesian coordinates in the 3D density map of each atom in the molecule. Figure 2 shows a sample interpreted density map. DEFT finds the coordinates of all atoms simultaneously by first building a pictorial structure corresponding to the protein, then using F-H matching to optimally place the model into the density map. This section describes DEFT's *deformation cost function* and *matching cost function*.

DEFT's *deformation cost* is related to the probability of observing a particular configuration of a molecule. Ideally, this function is proportional to the inverse of the molecule's potential function, since configurations with lower potential energy are more likely observed in nature. However, this potential is quite complicated and cannot be accurately approximated in a tree-structured pictorial structure graph.

Our solution is to *only consider the relationships between covalently bonded atoms*. DEFT constructs a pictorial structure graph where vertices correspond to non-hydrogen atoms, and edges correspond to the covalent bonds joining atoms. The cost function each edge defines maintain invariants – interatomic distance and bond angles – while allowing free rotation *around* the bond. Given the protein's amino acid sequence, model construction, illustrated in Figure 3, is trivial. Each part's configuration is defined by six parameters: three translational, three rotational (Euler angles $\alpha$, $\beta$, and $\gamma$). For the cost function, we define a new connection type in the pictorial structure framework, the *screw-joint*, shown in Figure 4.

The screw-joint's cost function is mathematically specified in terms of a directed version of the pictorial structure's undirected graph. Since the graph is constrained by the fast matching algorithm to take a tree structure, we arbitrarily pick a root node and point every edge toward this root. We now define the screw joint in terms of a *parent* and a *child*. We rotate the child such that its $z$ axis is coincident with the vector from child to parent, and allow each part in the model (that is, each atom) to freely rotate about its local $z$ axis. The ideal geometry between child and parent is then described by three parameters stored at each edge, $\mathbf{x_{ij}} = (x_{ij}, y_{ij}, z_{ij})$. These three parameters define the optimal translation between parent and child, in *the coordinate system of the parent* (which in turn is defined such that its $z$-axis corresponds to the axis connecting it to *its* parent).

In using these to construct the cost function $\mathbf{d_{ij}}$, we define the function $\mathbf{T_{ij}}$, which maps a parent $v_i$'s configuration $l_i$ into the configuration $l_j$ of that parent's *ideal child* $v_j$. Given parameters $\mathbf{x_{ij}}$ on the edge between $v_i$ and $v_j$, the function is defined

$$\mathbf{T_{ij}}\left(\langle x_i, y_i, z_i, \alpha_i, \beta_i, \gamma_i \rangle\right) = \langle x_j, y_j, z_j, \alpha_j, \beta_j, \gamma_j \rangle \tag{2}$$

with

$$\alpha_j = \alpha_i, \ \beta_j = \mathrm{atan2}\left(\sqrt{x'^2 + y'^2}, -z'\right), \ \gamma_j = \pi/2 + \mathrm{atan2}(y', x'), \text{ and}$$
$$\langle x_j, y_j, z_j \rangle = \langle x_i, y_i, z_i \rangle + \langle x', y', z' \rangle$$

where $(x', y', z')$ is rotation of the bond parameters $(x_{ij}, y_{ij}, z_{ij})$ to world coordinates. That is, $(x', y', z')^\mathsf{T} = \mathbf{R}_{\alpha_i, \beta_i, \gamma_i}(x_{ij}, y_{ij}, z_{ij})^\mathsf{T}$ with $\mathbf{R}_{\alpha_i, \beta_i, \gamma_i}$ the rotation matrix corresponding to Euler angles $(\alpha_i, \beta_i, \gamma_i)$. The expressions for $\beta_j$ and $\gamma_j$ define the optimal orientation of each child: $+z$ coincident with the axis that connects child and parent.

The F-H matching algorithm requires our cost function to take a particular form, specifically, it must be some norm. The screw-joint model sets the deformation cost between parent $v_i$ and child $v_j$ to the distance between child configuration $l_j$ and $\mathbf{T_{ij}}(l_i)$, the *ideal child configuration given parent configuration* $l_i$ ($\mathbf{T_{ji}}$ in equation **(2)** is simply the identity function). We use the 1-norm weighted in each dimension,

$$
\begin{aligned}
\mathbf{d_{ij}}(l_i, l_j) &= \left\| \mathbf{T_{ij}}(l_i) - l_j \right\| \\
&= w_{ij}^{rotate} \ \left| (\alpha_i - \alpha_j) \right| \\
&\quad + w_{ij}^{orient} \ \left( \left| (\beta_i - \beta_j) + \mathrm{atan}(\sqrt{x'^2 + y'^2}, -z') \right| + \left| (\gamma_j - \gamma_i) - \pi/2 + \mathrm{atan}(y', x') \right| \right) \\
&\quad + w_{ij}^{translate} \ \left( \left| (x_i - x_j) - x' \right| + \left| (y_i - y_j) - y' \right| + \left| (z_i - z_j) - z' \right| \right)
\end{aligned} \tag{3}
$$

In the above equation, $w_{ij}^{rotate}$ is the cost of rotating about a bond, $w_{ij}^{orient}$ is the cost of rotating around any other axis, and $w_{ij}^{translate}$ is the cost of translating in $x$, $y$ or $z$. DEFT's screw-joint model sets $w_{ij}^{rotate}$ to **0**, and $w_{ij}^{orient}$ and $w_{ij}^{translate}$ to **+100**.

DEFT's *match-cost function* implementation is based upon Cowtan's *fffear* algorithm [4]. This algorithm quickly and efficiently calculates the mean squared distance between a *weighted* 3D template of density and a region in a density map. Given a learned template and a corresponding *weight function*, *fffear* uses a Fourier convolution to determine the maximum likelihood that the weighted template generated a region of density in the density map.

For each non-hydrogen atom in the protein, we create a target template corresponding to a neighborhood around that particular atom, using a training set of crystallographer-solved structures. We build a separate template for each atom type – e.g., the β-carbon (2nd sidechain carbon) of leucine and the backbone oxygen of serine – producing 171 different templates in total. A part's $\mathbf{m}$ function is the *fffear*-computed mismatch score of that part's template over all positions and orientations.

Once we construct the model, parameters – including the optimal orientation $\mathbf{x_{ij}}$ corresponding to each edge, and the template for each part – are learned by training

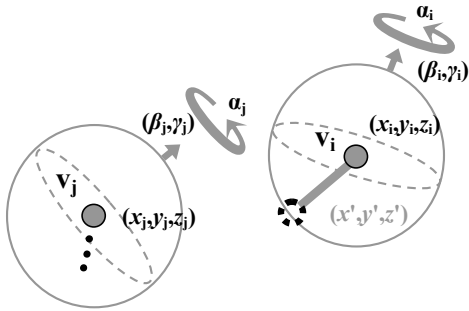

**Figure 4:** Showing the screw-joint connection between two parts in the model. In the directed version of the MRF, $v_i$ is the parent of $v_j$. By definition, $v_j$ is oriented such that its local $z$-axis is coincident with it's ideal bond orientation $\bar{\mathbf{x}}_{ij} = (x_{ij}, y_{ij}, z_{ij})^\mathsf{T}$ in $v_i$. Bond parameters $\bar{\mathbf{x}}_{ij}$ are learned by DEFT.

the model on a set of crystallographer-determined structures. Learning the orientation parameters is fairly simple: for each atom we define *canonic coordinates* (where *+z* corresponds to the axis of rotation). For each *child*, we record the distance *r* and orientation $(\theta, \varphi)$ *in the canonic coordinate frame*. We average over all atoms *of a given type* in our training set – e.g., over all leucine β-carbon's – to determine average parameters $r_{avg}$, $\theta_{avg}$, and $\varphi_{avg}$. Converting these averages from spherical to Cartesian coordinates gives the ideal orientation parameters $\mathbf{x_{ij}}$.

A similarly-defined canonic coordinate frame is employed when learning the model templates; in this case, DEFT's learning algorithm computes *target* and *weight* templates based on the average and inverse variance over the training set, respectively. Figure 5 shows an overview of the learning process. Implementation used Cowtan's *Clipper* library.

For each part in the model, DEFT searches through a six-dimensional conformation space $(x,y,z,\alpha,\beta,\gamma)$, breaking each dimension into a number of discrete bins. The translational parameters *x*, *y*, and *z* are sampled over a region in the unit cell. Rotational space is *uniformly* sampled using an algorithm described by Mitchell [6].

## 4   Model Enhancements

Upon initial testing, the pictorial-structure matching algorithm performs rather poorly at the density-map interpretation task. Consequently, we added two routines – a collision-detection routine, and an improved template-matching routine – to DEFT's pictorial-structure matching implementation. Both enhancements can be applied to the general pictorial structure algorithm, and are not specific to DEFT.

### 4.1   Collision Detection

Our closer investigation revealed that much of the algorithm's poor performance is due to distant chains *colliding*. Since DEFT *only models covalent bonds*, the matching algorithm sometimes returns a structure with non-bonded atoms impossibly close together. These collisions were a problem in DEFT's initial implementation. Figure 6 shows such a collision (later corrected by the algorithm).

Given a candidate solution, it is straightforward to test for spatial collisions: we simply test if any two atoms in the structure are impossibly (physically) close together. If a collision occurs in a candidate, DEFT perturbs the structure. Though

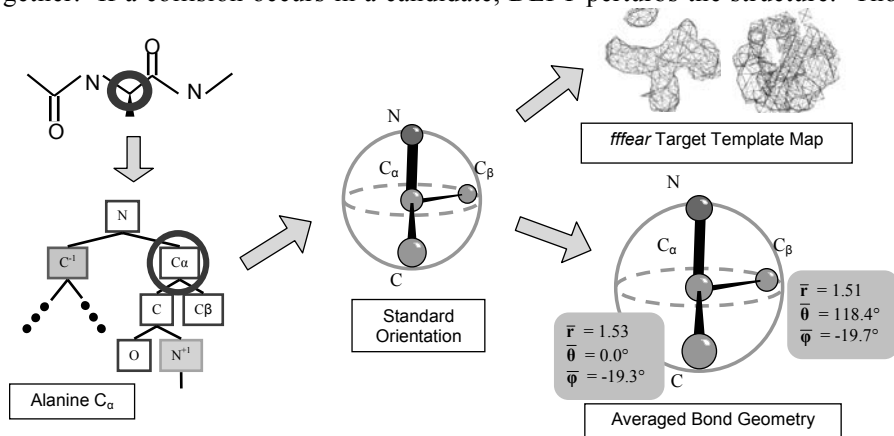

**Figure 5:** An overview of the parameter-learning process. For each atom of a given type – here alanine $C_\alpha$ – we rotate the atom into a canonic orientation. We then average over every atom of that type to get a template and average bond geometry.

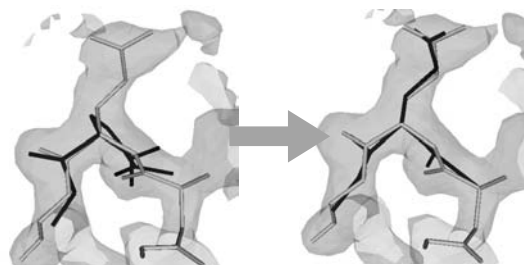

**Figure 6.** This illustrates the collision avoidance algorithm. On the left is a *collision* (the predicted molecule is in the darker color). The amino acid's sidechain is placed coincident with the back-bone. On the right, collision avoidance finds the right structure.

the optimal match is no longer returned, this approach works well in practice. If two atoms are both aligned to the same space in the most probable conformation, it seems quite likely that *one* of the atoms belongs there. Thus, DEFT handles collisions by assuming that at least one of the two colliding branches is correct. When a collision occurs, DEFT finds the closest branch point above the colliding nodes – that is, the root $y$ of the minimum subtree containing all colliding nodes. DEFT considers each *child $x_i$* of this root, matching the subtree rooted at $x_i$, keeping the remainder of the tree fixed. The change in score for each perturbed branch is recorded, and the one with the smallest score increase is the one DEFT keeps.

Table 1 describes the collision-avoidance algorithm. In the case that the colliding node is due to a chain wrapping around on itself (and not two branches running into one another), the root $y$ is defined as the colliding node nearest to the top of the tree. Everything below $y$ is matched anew while the remainder of the structure is fixed.

## 4.2   Improved template matching

In our original implementation, DEFT learned a template by averaging over each of the 171 atom types. For example, for each of the 12 (non-hydrogen) atoms in the amino-acid tyrosine we build a single template – producing 12 tyrosine templates in total. Not only is this inefficient, requiring DEFT to match redundant templates against the unsolved density map, but also for some atoms in flexible sidechains, averaging blurs density contributions from atoms more than a bond away from the target, losing valuable information about an atom's neighborhood.

DEFT improves the template-matching algorithm by modeling the templates using a *mixture of Gaussians*, a generative model where each template is modeled using a mixture of basis templates. Each basis template is simply the mean of a cluster of templates. Cluster assignments are learned iteratively using the EM algorithm. In each iteration of the algorithm we compute the *a priori* likelihood of each image being generated by a particular cluster mean (the E step). Then we use these probabilities to update the cluster means (the M step). After convergence, we use each cluster mean (and weight) as an *fffear* search target.

**Table 1**. DEFT's collision handing routine.

---

**Given:**    An illegal pictorial structure configuration $L = \{l_1, l_2, \ldots, l_n\}$
**Return:**   A legal perturbation $L'$

**Algorithm:**

    $X \leftarrow$ all nodes in $L$ illegally close to some other node
    $y \leftarrow$ root of smallest subtree containing all nodes in $X$

    for each child $x_i$ of $y$
        $L_i \leftarrow$ optimal position of subtree rooted at $x_i$ *fixing remainder of tree*
        $score_i \leftarrow \textbf{score}(L_i) - \textbf{score}$(subtree of $L$ rooted at $x_i$)

    $i_{min} \leftarrow$ arg min ($score_i$)
    $L' \leftarrow$ replace subtree rooted at $x_i$ in $L$ with $L_{imin}$

    return $L'$

---

# 5  Experimental Studies

We tested DEFT on a set of proteins provided by the Phillips lab at the University of Wisconsin. The set consists of four different proteins, all around 2.0Å in resolution. With all four proteins, reflections and experimentally-determined initial phases were provided, allowing us to build four relatively poor-quality density maps. To test our algorithm with poor-quality data, we down-sampled each of the maps to 2.5, 3 and 4Å by removing higher-resolution reflections and recomputed the density. These down-sampled maps are physically identical to maps natively constructed at this resolution. Each structure had been solved by crystallographers.

For this paper, our experiments are conducted under the assumption that the mainchain atoms of the protein were known to within some error factor. This assumption is fair; approaches exist for mainchain tracing in density maps [7]. DEFT simply walks along the mainchain, placing atoms one residue at a time (considering each residue independently).

We split our dataset into a training set of about 1000 residues and a test set of about 100 residues (from a protein **not** in the training set). Using the training set we built a set of templates for matching using *fffear*. The templates extended to a 6Å radius around each atom at 0.5Å sampling. Two sets of templates were built and subsequently matched: a large set of 171 produced by averaging all training set templates for each atom type, and a smaller set of 24 learned through by the EM algorithm. We ran DEFT's pictorial structure matching algorithm using both sets of templates, with and without the collision detection code.

Although placing individual atoms into the sidechain is fairly quick, taking less than six hours for a 200-residue protein, computing *fffear* match scores is very CPU-demanding. For each of our 171 templates, *fffear* takes 3-5 CPU-hours to compute the match score at each location in the image, for a total of one CPU-month to match templates into each protein! Fortunately the task is trivially parallelized; we regularly do computations on over 100 computers simultaneously.

The results of all tests are summarized in Figure 7. Using individual-atom templates and the collision detection code, the all-atom RMS deviation varied from 1.38Å at 2Å resolution to 1.84Å at 4Å. Using the EM-based clusters as templates produced slight or no improvement. However, much less work is required; only 24 templates need to be matched to the image instead of 171 individual-atom templates. Finally, it was promising that collision detection leads to significant error reduction.

It is interesting to note that individually using the improved templates and using the collision avoidance both improved the search results; however, using both together was a bit worse than with collision detection alone. More research is needed to get a synergy between the two enhancements. Further investigation is also needed balancing between the number and templates and template size. The match cost function is a critically important part of DEFT and improvements there will have the most profound impact on the overall error.

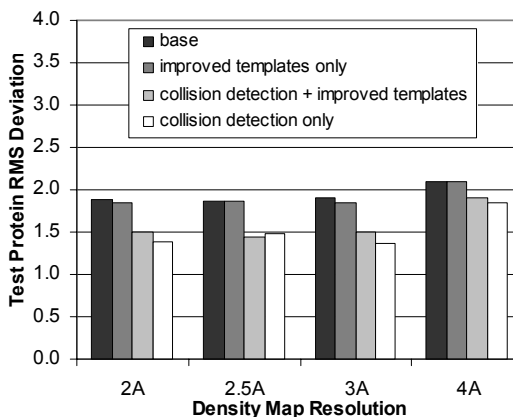

**Figure 7.** Testset error under four strategies.

# 6 Conclusions and future work

DEFT has applied the F-H pictorial structure matching algorithm to the task of interpreting electron density maps. In the process, we extended the F-H algorithm in three key ways. In order to model atoms rotating in 3D, we designed another joint type, the screw joint. We also developed extensions to deal with spatial collisions of parts in the model, and implemented a slightly-improved template construction routine. Both enhancements can be applied to pictorial-structure matching in general, and are not specific to the task presented here.

DEFT attempts to bridge the gap between two types of model-fitting approaches for interpreting electron density maps. Several techniques [1,2,3] do a good job placing individual atoms, but all fail around 2.5-3Å resolution. On the other hand, *fffear* [4] has had success finding *rigid elements* in very poor resolution maps, but is unable to locate highly flexible "loops". Our work extends the resolution threshold at which individual atoms can be identified in electron density maps. DEFT's flexible model combines weakly-matching image templates to locate individual atoms from maps where individual atoms have been blurred away. No other approach has investigated sidechain refinement in structures of this poor resolution.

We next plan to use DEFT as the refinement phase complementing a coarser method. Rather than model the configuration of each individual atom, instead treat each amino acid as a single part in the flexible template, only modeling rotations *along* the backbone. Then, our current algorithm could place each individual atom.

A different optimization algorithm that handles cycles in the pictorial structure graph would better handle collisions (allowing edges between non-bonded atoms). In recent work [8], loopy belief propagation [9] has been used with some success (though with no optimality guarantee). We plan to explore the use of belief propagation in pictorial-structure matching, adding edges in the graph to avoid collisions.

Finally, the pictorial-structure framework upon which DEFT is built seems quite robust; we believe the accuracy of our approach can be substantially improved through implementation improvements, allowing finer grid spacing and larger *fffear* ML templates. The flexible molecular template we have described has the potential to produce an atomic model in a map where individual atoms may not be visible, through the power of combining weakly matching image templates. DEFT could prove important in high-throughput protein-structure determination.

## Acknowledgments

This work supported by NLM Grant 1T15 LM007359-01, NLM Grant 1R01 LM07050-01, and NIH Grant P50 GM64598.

## References

[1] A. Perrakis, T. Sixma, K. Wilson, & V. Lamzin (1997). wARP: improvement and extension of crystallographic phases. *Acta Cryst.* D53:448-455.

[2] D. Levitt (2001). A new software routine that automates the fitting of protein X-ray crystallographic electron density maps. *Acta Cryst.* D57:1013-1019.

[3] T. Ioerger, T. Holton, J. Christopher, & J. Sacchettini (1999). TEXTAL: a pattern recognition system for interpreting electron density maps. *Proc. ISMB*:130-137.

[4] K. Cowtan (2001). Fast fourier feature recognition. *Acta Cryst.* D57:1435-1444.

[5] P. Felzenszwalb & D. Huttenlocher (2000). Efficient matching of pictorial structures. *Proc. CVPR*. pp. 66-73.

[6] J. Mitchell (2002). Uniform distributions of 3D rotations. Unpublished Document.

[7] J. Greer (1974). Three-dimensional pattern recognition. *J. Mol. Biol.* 82:279-301.

[8] E. Sudderth, M. Mandel, W. Freeman & A Willsky (2005). Distributed occlusion reasoning for tracking with nonparametric belief propagation. *NIPS*.

[9] D. Koller, U. Lerner & D. Angelov (1999). A general algorithm for approximate inference and its application to hybrid Bayes nets. *UAI.* 15:324-333.
